# A Trellis-Structured Neural Network*

Thomas Petsche† and Bradley W. Dickinson
Princeton University, Department of Electrical Engineering
Princeton, NJ 08544

### Abstract

We have developed a neural network which consists of cooperatively inter-connected Grossberg on-center off-surround subnets and which can be used to optimize a function related to the log likelihood function for decoding convolutional codes or more general FIR signal deconvolution problems. Connections in the network are confined to neighboring subnets, and it is representative of the types of networks which lend themselves to VLSI implementation. Analytical and experimental results for convergence and stability of the network have been found. The structure of the network can be used for distributed representation of data items while allowing for fault tolerance and replacement of faulty units.

## 1  Introduction

In order to study the behavior of locally interconnected networks, we have focused on a class of "trellis-structured" networks which are similar in structure to multilayer networks [5] but use symmetric connections and allow every neuron to be an output. We are studying such locally interconnected neural networks because they have the potential to be of great practical interest. Globally interconnected networks, e.g., Hopfield networks [3], are difficult to implement in VLSI because they require many long wires. Locally connected networks, however, can be designed to use fewer and shorter wires.

In this paper, we will describe a subclass of trellis-structured networks which optimize a function that, near the global minimum, has the form of the log likelihood function for decoding convolutional codes or more general finite impulse response signals. Convolutional codes, defined in section 2, provide an alternative representation scheme which can avoid the need for global connections. Our network, described in section 3, can perform maximum likelihood sequence estimation of convolutional coded sequences in the presence of noise. The performance of the system is optimal for low error rates.

The specific application for this network was inspired by a signal decomposition network described by Hopfield and Tank [6]. However, in our network, there is an emphasis on local interconnections and a more complex neural model, the Grossberg on-center off-surround network [2], is used. A modified form of the Gorssberg model is defined in section 4. Section 5 presents the main theoretical results of this paper. Although the deconvolution network is simply a set of cooperatively interconnected

*Supported by the Office of Naval Research through grant N00014-83-K-0577 and by the National Science Foundation through grant ECS84-05460.

†Permanent address: Siemens Corporate Research and Support, Inc., 105 College Road East, Princeton, NJ 08540.

on-center off-surround subnetworks, and absolute stability for the individual subnetworks has been proven [1], the cooperative interconnections between these subnets make a similar proof difficult and unlikely. We have been able, however, to prove equiasymptotic stability in the Lyapunov sense for this network given that the gain of the nonlinearity in each neuron is large. Section 6 will describe simulations of the network that were done to confirm the stability results.

## 2 Convolutional Codes and MLSE

In an error correcting code, an input sequence is transformed from a $b$-dimensional input space to an $M$-dimensional output space, where $M \geq b$ for error correction and/or detection. In general, for the $b$-bit input vector $\mathbf{U} = (u_1, \ldots, u_b)$ and the $M$-bit output vector $\mathbf{V} = (v_1, \ldots, v_M)$, we can write $\mathbf{V} = F(u_1, \ldots, u_b)$. A convolutional code, however, is designed so that relatively short subsequences of the input vector are used to determine subsequences of the output vector. For example, for a rate $1/3$ convolutional code (where $M \approx 3b$), with input subsequences of length 3, we can write the output, $\mathbf{V} = (\mathbf{v}_1, \ldots, \mathbf{v}_b)$ for $\mathbf{v}_i = (v_{i,1}, v_{i,2}, v_{i,3})$, of the encoder as a convolution of the input vector $\mathbf{U} = (u_1, \ldots, u_b, 0, 0)$ and three generator sequences

$$\mathbf{g}_0 = (1\ 1\ 1) \qquad \mathbf{g}_1 = (1\ 1\ 0) \qquad \mathbf{g}_2 = (0\ 1\ 1).$$

This convolution can be written, using modulo-2 addition, as

$$\mathbf{v}_i = \sum_{k=\max(1,i-2)}^{i} u_k \mathbf{g}_{i-k} \qquad (1)$$

In this example, each 3-bit output subsequence, $\mathbf{v}_i$, of $\mathbf{V}$ depends only on three bits of the input vector , i.e., $\mathbf{v}_i = f(u_{i-2}, u_{i-1}, u_i)$. In general, for a rate $1/n$ code, the *constraint length*, $K$, is the number of bits of the input vector that uniquely determine each $n$-bit output subsequence. In the absence of noise, any subsequences in the input vector separated by more than $K$ bits (i.e., that do not overlap) will produce subsequences in the output vector that are independent of each other.

If we view a convolutional code as a special case of block coding, this rate $1/3$, $K = 3$ code converts a $b$-bit input word into a codeword of length $3(b + 2)$ where the 2 is added by introducing two zeros at the end of every input to "zero-out" the code. Equivalently, the coder can be viewed as embedding $2^b$ memories into a $2^{3(b+2)}$-dimensional space. The minimum distance between valid memories or codewords in this space is the *free distance* of the code, which in this example is 7. This implies that the code is able to correct a minimum of three errors in the received signal.

For a convolutional code with constraint length $K$, the encoder can be viewed as a finite state machine whose state at time $i$ is determined by the $K - 1$ input bits, $u_{i-k}, \ldots, u_{i-1}$. The encoder can also be represented as a trellis graph such as the one shown in figure 1 for a $K = 3$, rate $1/3$ code. In this example, since the constraint length is three, the two bits $u_{i-2}$ and $u_{i-1}$ determine which of four possible states the encoder is in at time $i$. In the trellis graph, there is a set of four nodes arranged in a vertical column, which we call a stage, for each time step $i$. Each node is labeled with the associated values of $u_{i-2}$ and $u_{i-1}$. In general, for a rate $1/n$ code, each stage of the trellis graph contains $2^{K-1}$ nodes, representing an equal number of possible states. A trellis graph which contains $S$ stages therefore fully describes the operation of the encoder for time steps 1 through $S$. The graph is read from left to right and the upper edge leaving the right side of a node in stage $i$ is followed if $u_i$ is a zero; the lower edge

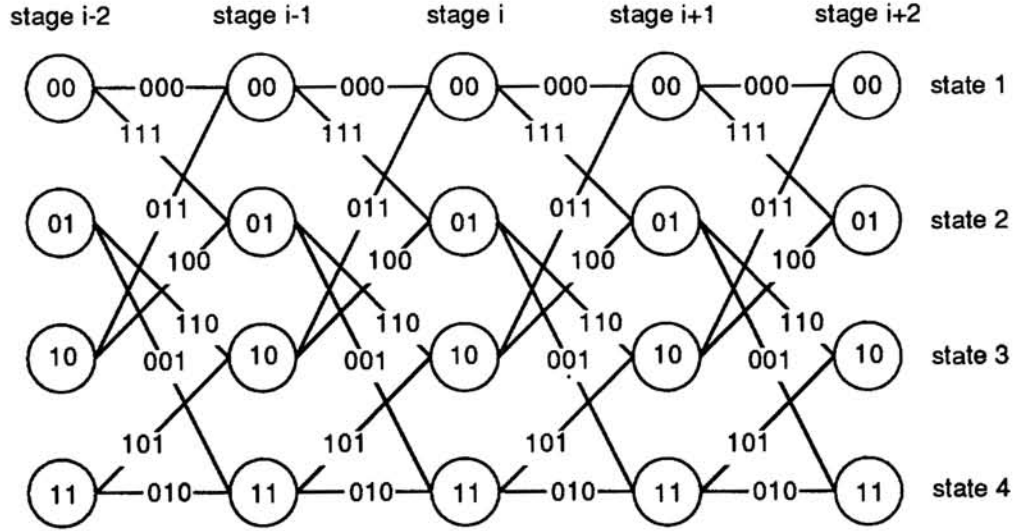

Figure 1: Part of the trellis-code representation for a rate $1/3$, $K = 3$ convolutional code.

if $u_i$ is a one. The label on the edge determined by $u_i$ is $v_i$, the output of the encoder given by equation 1 for the subsequence $u_{i-2}$, $u_{i-1}$, $u_i$.

Decoding a noisy sequence that is the output of a convolutional coder plus noise is typically done using a maximum likelihood sequence estimation (MLSE) decoder which is designed to accept as input a possibly noisy convolutional coded sequence, $\mathbf{R}$, and produce as output the maximum likelihood estimate, $\hat{\mathbf{V}}$, of the original sequence, $\mathbf{V}$. If the set of possible $n(b+2)$-bit encoder output vectors is $\{\mathbf{X}_m : m = 1, ..., 2^{n(b+2)}\}$ and $\mathbf{x}_{m,i}$ is the $i$th $n$-bit subsequence of $\mathbf{X}_m$ and $\mathbf{r}_i$ is the $i$th $n$-bit subsequence of $\mathbf{R}$ then

$$\hat{\mathbf{V}} = \arg\max_{\mathbf{X}_m} \prod_{i=1}^{b} \mathrm{P}(\mathbf{r}_i \mid \mathbf{x}_{m,i}) \tag{2}$$

That is, the decoder chooses the $\mathbf{X}_m$ that maximizes the conditional probability, given $\mathbf{X}_m$, of the received sequence.

A binary symmetric channel (BSC) is an often used transmission channel model in which the decoder produces output sequences formed from an alphabet containing two symbols and it is assumed that the probability of either of the symbols being affected by noise so that the other symbol is received is the same for both symbols. In the case of a BSC, the log of the conditional probability, $\mathrm{P}(\mathbf{r}_i \mid \mathbf{x}_{m,i})$, is a linear function of the Hamming distance between $\mathbf{r}_i$ and $\mathbf{x}_{m,i}$ so that maximizing the right side of equation 2 is equivalent to choosing the $\mathbf{X}_m$ that has the most bits in common with $\mathbf{R}$. Therefore, equation 2 can be rewritten as

$$\hat{\mathbf{V}} = \arg\max_{\mathbf{X}_m} \sum_{i=1}^{b} \sum_{l=1}^{n} I_{r_{i,l}}(x_{m,i,l}) \tag{3}$$

where $\mathbf{x}_{m,i,l}$ is the $l$th bit of the $i$th subsequence of $\mathbf{X}_m$ and $I_a(b)$ is the indicator function: $I_a(b) = 1$ if and only if $a$ equals $b$.

For the general case, maximum likelihood sequence estimation is very expensive since the number of possible input sequences is exponential in $b$. The Viterbi algorithm [7], fortunately, is able to take advantage of the structure of convolutional codes and their trellis graph representations to reduce the complexity of the decoder so that

it is only exponential in $K$ (in general $K \ll b$). An optimum version of the Viterbi algorithm examines all $b$ stages in the trellis graph, but a more practical and very nearly optimum version typically examines approximately $5K$ stages, beginning at stage $i$, before making a decision about $u_i$.

# 3  A Network for MLSE Decoding

The structure of the network that we have defined strongly reflects the structure of a trellis graph. The network usually consists of $5K$ subnetworks, each containing $2^{K-1}$ neurons. Each subnetwork corresponds to a stage in the trellis graph and each neuron to a state. Each stage is implemented as an "on-center off-surround" competitive network [2], described in more detail in the next section, which produces as output a contrast enhanced version of the input. This contrast enhancement creates a "winner take all" situation in which, under normal circumstances, only one neuron in each stage —the neuron receiving the input with greatest magnitude — will be on. The activation pattern of the network after it reaches equilibrium indicates the decoded sequence as a sequence of "on" neurons in the network. If the $j$-th neuron in subnet $i$, $\mathcal{N}_{i,j}$ is on, then the node representing state $j$ in stage $i$ lies on the network's estimate of the most likely path.

For a rate $1/n$ code, there is a symmetric cooperative connection between neurons $\mathcal{N}_{i,j}$ and $\mathcal{N}_{i+1,k}$ if there is an edge between the corresponding nodes in the trellis graph. If $(x_{i,j,k,1}, \ldots, x_{i,j,k,n})$ are the encoder output bits for the transition between these two nodes and $(r_{i,1}, \ldots, r_{i,n})$ are the received bits, then the connection weight for the symmetric cooperative connection between $\mathcal{N}_{i,j}$ and $\mathcal{N}_{i+1,k}$ is

$$m_{i,j,k} = \frac{1}{n} \sum_{l=1}^{n} I_{r_{i,l}}(x_{i,j,k,l}) \tag{4}$$

If there is no edge between the nodes, then $m_{i,j,k} = 0$.

Intuitively, it is easiest to understand the action of the entire network by examining one stage. Consider the nodes in stage $i$ of the trellis graph and assume that the conditional probabilities of the nodes in stages $i-1$ and $i+1$ are known. (All probabilities are conditional on the received sequence.) Then the conditional probability of each node in stage $i$ is simply the sum of the probabilities of each node in stages $i-1$ and $i+1$ weighted by the conditional transition probabilities. If we look at stage $i$ in the network, and let the outputs of the neighboring stages $i-1$ and $i+1$ be fixed with the output of each neuron corresponding to the "likelihood" of the corresponding state at that stage, then the final outputs of the neurons $\mathcal{N}_{i,j}$ will correspond to the "likelihood" of each of the corresponding states. At equilibrium, the neuron corresponding to the most likely state will have the largest output.

# 4  The Neural Model

The "on-center off-surround" network[2] is used to model each stage in our network. This model allows the output of each neuron to take on a range of values, in this case between zero and one, and is designed to support contrast enhancement and competition between neurons. The model also guarantees that the final output of each neuron is a function of the relative intensity of its input as a fraction of the total input provided to the network.

Using the "on-center off-surround" model for each stage and the interconnection weights, $m_{i,j,k}$, defined in equation 4, the differential equation that governs the instantaneous activity of the neurons in our deconvolution network with $S$ stages and $N$ states in each stage can be written as

$$\dot{u}_{i,j} = -Au_{i,j} + (B - u_{i,j})\left(f(u_{i,j}) + \sum_{k=1}^{N}[m_{i-1,k,j}f(u_{i-1,k}) + m_{i,j,k}f(u_{i+1,k})]\right)$$
$$- (C + u_{i,j})\sum_{k \neq j}^{N}\left(f(u_{i,k}) + \sum_{l=1}^{N}[m_{i-1,k,l}f(u_{i-1,k}) + m_{i,l,k}f(u_{i+1,k})]\right) \quad (5)$$

where $f(x) = (1 + e^{-\lambda x})^{-1}$, $\lambda$ is the gain of the nonlinearity, and $A$, $B$, and $C$ are constants

For the analysis to be presented in section 5, we note that equation 5 can be rewritten more compactly in a notation that is similar to the equation for additive analog neurons given in [4]:

$$\dot{u}_{i,j} = -Au_{i,j} - \sum_{k=1}^{S}\sum_{l=1}^{N}(u_{i,j}S_{i,j,k,l}f(u_{k,l}) - T_{i,j,k,l}f(u_{k,l})) \quad (6)$$

where, for $1 \leq l \leq N$,

$$\begin{aligned} S_{i,j,i,l} &= 1 \\ S_{i,j,i-1,l} &= \sum_{q} m_{i-1,l,q} \\ S_{i,j,i+1,l} &= \sum_{q} m_{i,q,l} \\ S_{i,j,k,l} &= 0 \quad \forall k \notin \{i-1, i, i+1\} \end{aligned} \qquad \begin{aligned} T_{i,j,i,j} &= B \\ T_{i,j,i,l} &= -C \quad \forall l \neq j \\ T_{i,j,i-1,l} &= Bm_{i-1,l,j} - C\sum_{q \neq j} m_{i-1,l,q} \\ T_{i,j,i+1,l} &= Bm_{i,j,l} - C\sum_{q \neq j} m_{i,q,l} \end{aligned} \quad (7)$$

To eliminate the need for global interconnections within a stage, we can add two summing elements to calculate

$$X_i = \sum_{j=1}^{N} f(x_{i,j}) \quad \text{and} \quad J_i = \sum_{j=1}^{N}\sum_{k=1}^{N}[m_{i-1,k,j}f(u_{i-1,k}) + m_{i,j,k}f(u_{i+1,k})] \quad (8)$$

Using these two sums allows us to rewrite equation 5 as

$$\dot{u}_{i,j} = -Au_{i,j} + (B + C)(f(u_{i,j}) + I_{i,j}) - u_{i,j}(X_i + J_i) \quad (9)$$

This form provides a more compact design for the network that is particularly suited to implementation as a digital filter or for use in simulations since it greatly reduces the calculations required.

## 5  Stability of the Network

The end of section 3 described the desired operation of a single stage, given that the outputs of the neighboring stages are fixed. It is possible to show that in this situation a single stage is stable. To do this, fix $f(u_{k,l})$ for $k \in \{i-1, i+1\}$ so that equation 6 can be written in the form originally proposed by Grossberg [2]:

$$\dot{u}_{i,j} = -Au_{i,j} + (B - u_{i,j})(I_{i,j} + f(u_{i,j})) - (u_{i,j} + C)\left(\sum_{k=1}^{N} I_{i,k} + \sum_{k=1}^{N} f(u_{i,k})\right) \quad (10)$$

where $I_{i,j} = \sum_{k=1}^{N} [m_{i-1,k,j} f(u_{i-1,k}) + m_{i,j,k} f(u_{i+1,k})]$.

Equation 10 is a special case of the more general nonlinear system

$$\dot{x}_i = a_i(x_i)\left(b_i(x_i) - \sum_{k=1}^{n} c_{i,k} d_k(x_k)\right) \tag{11}$$

where: (1) $a_i(x_i)$ is continuous and $a_i(x_i) > 0$ for $x_i \geq 0$; (2) $b_i(x_i)$ is continuous for $x_i \geq 0$; (3) $c_{i,k} = c_{k,i}$; and (4) $d_i(x_i) \geq 0$ for all $x_i \in (-\infty, \infty)$. Cohen and Grossberg [1] showed that such a system has a global Lyapunov function:

$$V(\mathbf{x}) = -\sum_{i=1}^{n} \int_0^{x_i} b_i(\xi_i) d_i'(\xi_i) d(\xi_i) + \frac{1}{2} \sum_{j=1}^{n} \sum_{k=1}^{n} c_{j,k} d_j(x_j) d_k(x_k) \tag{12}$$

and that, therefore, such a system is equiasymptotically stable for all constants and functions satisfying the four constraints above. In our case, this means that a single stage has the desired behavior when the neighboring stages are fixed. If we take the output of each neuron to correspond to the likelihood of the corresponding state then, if the two neighboring stages are fixed, stage $i$ will converge to an equilibrium point where the neuron receiving the largest input will be on and the others will be off, just as it should according to section 2.

It does not seem possible to use the Cohen-Grossberg stability proof for the entire system in equation 5. In fact, Cohen and Grossberg note that networks which allow cooperative interactions define systems for which no stability proof exists [1].

Since an exact stability proof seems unlikely, we have instead shown that in the limit as the gain, $\lambda$, of the nonlinearity gets large the system is asymptotically stable. Using the notation in [4], define $V_i = f(u_i)$ and a normalized nonlinearity $\bar{f}(\cdot)$ such that $\bar{f}^{-1}(V_i) = \lambda u_i$. Then we can define an energy function for the deconvolution network to be

$$E = -\frac{1}{2} \sum_{i,j,k,l} T_{i,j,k,l} V_{i,j} V_{k,l} - \sum_{i,j} \frac{1}{\lambda}\left(-A - \sum_{k,l} S_{i,j,k,l} V_{k,l}\right) \int_{\frac{1}{2}}^{V_{k,l}} \bar{f}^{-1}(\zeta)\, d\zeta \tag{13}$$

The time derivative of $E$ is

$$\dot{E} = -\sum_{i,j} \frac{dV_{i,j}}{dt}\left(-A u_{i,j} - u_{i,j} \sum_{k,l} S_{i,j,k,l} V_{k,l} + \sum_{k,l} T_{i,j,k,l} V_{k,l} \right.$$
$$\left. -\frac{1}{\lambda} \sum_{k,l} S_{i,j,k,l} \int_{\frac{1}{2}}^{V_{k,l}} \bar{f}^{-1}(\zeta) d\zeta\right) \tag{14}$$

It is difficult to prove that $\dot{E}$ is nonpositive because of the last term in the parentheses. However, for large gain, this term can be shown to have a negligible effect on the derivative.

It can be shown that for $f(u) = (1 + e^{-\lambda u})^{-1}$, $\int_{\frac{1}{2}}^{V_i} \bar{f}_i^{-1}(\zeta)\, d\zeta$ is bounded above by $\log(2)$. In this deconvolution network, there are no connections between neurons unless they are in the same or neighboring stages, i.e., $S_{i,j,k,l} = 0$ for $|i - k| > 1$ and $l$ is restricted so that $0 \leq l \leq S$, so there are no more than $3S$ non-zero terms in the problematical summation. Therefore, we can write that

$$\lim_{\lambda \to \infty} -\frac{1}{\lambda} \sum_{k,l} S_{i,j,k,l} \int_{\frac{1}{2}}^{V_{k,l}} \bar{f}^{-1}(\zeta)\, d\zeta = 0$$

Then, in the limit as $\lambda \rightarrow \infty$, the terms in parentheses in equation 14 converge to $\dot{u}_i$ in equation 6, so that $\lim_{\lambda \rightarrow \infty} \dot{E} = \sum_{i,j} \frac{dV_{i,j}}{dt} \dot{u}_i$. Using the chain rule, we can rewrite this as

$$\lim_{\lambda \rightarrow \infty} \dot{E} = -\sum_{i,j} \left( \frac{dV_{i,j}}{dt} \right)^2 \left( \frac{d}{dV_{i,j}} f^{-1}(V_{i,j}) \right) \quad .$$

It can also be shown that that, if $f(\cdot)$ is a monotonically increasing function then $\frac{d}{dV_i} f^{-1}(V_i) > 0$ for all $V_i$. This implies that for all $\mathbf{u} = (u_{i,1}, \ldots, u_{N,S})$, $\lim_{\lambda \rightarrow \infty} \dot{E} \leq 0$, and, therefore, for large gains, $E$ as defined in equation 13 is a Lyapunov function for the system described by equation 5 and the network is equiasymtotically stable.

If we apply a similar asymptotic argument to the energy function, equation 13 reduces to

$$E = -\frac{1}{2} \sum_{i,j,k,l} T_{i,j,k,l} V_{i,j} V_{k,l} \qquad (15)$$

which is the Lyapunov function for a network of discontinuous on-off neurons with interconnection matrix $\mathbf{T}$. For the binary neuron case, it is fairly straight forward to show that the energy function has minima at the desired decoder outputs if we assume that only one neuron in each stage may be on and that $B$ and $C$ are appropriately chosen to favor this. However, since there are $O(S^2 N)$ terms in the disturbance summation in equation 15, convergence in this case is not as fast as for the derivative of the energy function in equation 13, which has only $O(S)$ terms in the summation.

## 6 Simulation Results

The simulations presented in this section are for the rate $1/3$, $K = 3$ convolutional code illustrated in figure 1. Since this code has a constraint length of 3, there are 4 possible states in each stage and an MLSE decoder would normally examine a minimum of $5K$ subsequences before making a decision, we will use a total of 16 stages. In these simulations, the first and last stage are fixed since we assume that we have prior knowledge or a decision about the first stage and zero knowledge about the last stage. The transmitted codeword is assumed to be all zeros.

The simulation program reads the received sequence from standard input and uses it to define the interconnection matrix $W$ according to equation 4. A relaxation subroutine is then called to simulate the performance of the network according to an Euler discretization of equation 5. Unit time is then defined as one RC time constant of the unforced system. All variables were defined to be single precision (32 bit) floating point numbers.

Figure 2a shows the evolution of the network over two unit time intervals with the sampling time $T = 0.02$ when the received codeword contains no noise. To interpret the figure, recall that there are 16 stages of 4 neurons each. The output of each stage is a vertical set of 4 curves. The upper-left set is the output of the first stage; the upper-most curve is the output of the first neuron in the stage. For the first stage, the first neuron has a fixed output of 1 and the other neurons have a fixed output of 0. The outputs of the neurons in the last stages are fixed at an intermediate value to represent zero *a priori* knowledge about these states. Notice that the network reaches an equilibrium point in which only the top neurons in each state (representing the "00" node in figure 1) are on and all others are off. This case illustrates that the network can correctly decode an unerrored input and that it does so rapidly, i.e., in about one time constant. In this case, with no errors in the input, the network performs the

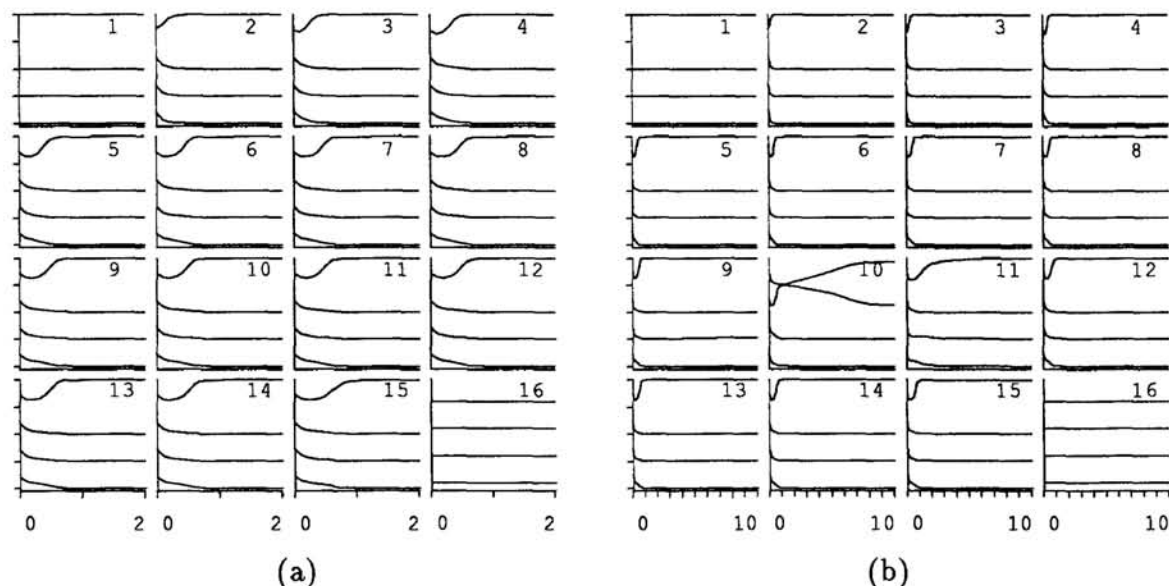

Figure 2: Evolution of the trellis network for (a) unerrored input, (b) input with burst errors: R is 000 000 000 000 000 000 000 000 111 000 000 000 000 000 000. $\lambda = 10.$, $A = 1.0$, $B = 1.0$, $C = 0.75$, $T = 0.02$. The initial conditions are $x_{1,1} = 1.$, $x_{1,j} = 0.0$, $x_{16,j} = 0.2$, all other $x_{i,j} = 0.0$.

same function as Hopfield and Tank's network and does so quite well. Although we have not been able to prove it analytically, all our simulations support the conjecture that if $x_{i,j}(0) = \frac{1}{2}$ for all $i$ and $j$ then the network will always converge to the global minimum.

One of the more difficult decoding problems for this network is the correction of a burst of errors in a transition subsequence. Figure 2b shows the evolution of the network when three errors occur in the transition between stages 9 and 10. Note that 10 unit time intervals are shown since complete convergence takes much longer than in the first example. However, the network has correctly decoded many of the stages far from the burst error in a much shorter time.

If the received codeword contains scattered errors, the convolutional decoder should be able to correct more than 3 errors. Such a case is shown in figure 3a in which the received codeword contains 7 errors. The system takes longest to converge around two transitions, 5-6 and 11-12. The first is in the midst of consecutive subsequences which each have one bit errors and the second transition contains two errors.

To illustrate that the energy function shown in equation 13 is a good candidate for a Lyapunov function for this network, it is plotted in figure 3b for the three cases described above. The nonlinearity used in these simulations has a gain of ten, and, as predicted by the large gain limit, the energy decreases monotonically.

To more thoroughly explore the behavior of the network, the simulation program was modified to test many possible error patterns. For one and two errors, the program exhaustively tested each possible error pattern. For three or more errors, the errors were generated randomly. For four or more errors, only those errored sequences for which the MLS estimate was the sequence of all zeros were tested. The results of this simulation are summarized in the column labeled "two-nearest" in figure 4. The performance of the network is optimum if no more than 3 errors are present in the received sequence, however for four or more errors, the network fails to correctly decode some sequences that the MLSE decoder can correctly decode.

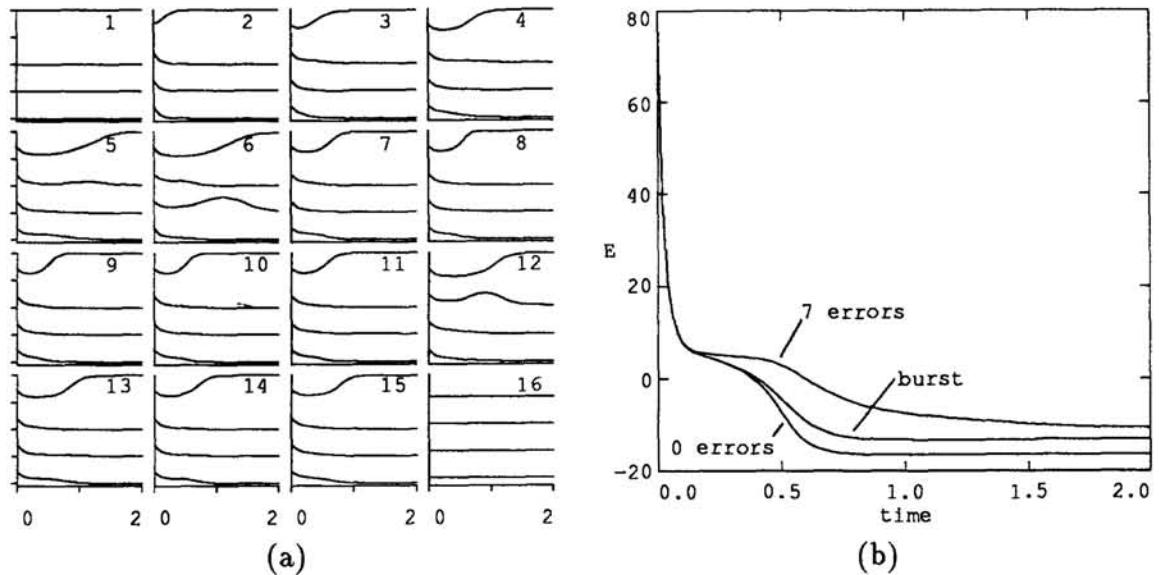

(a)                                    (b)

Figure 3: (a) Evolution of the trellis network for input with distributed errors. The input, **R**, is 000 010 010 010 100 001 000 000 000 000 110 000 000 000 000. The constants and initial conditions are the same as in figure 2. (b) The energy function defined in equation 13 evaulated for the three simulations discussed.

| errored bits | number of test vectors | number of errors | |
|---|---|---|---|
| | | two-nearest | four-nearest |
| 0 | 1 | 0 | 0 |
| 1 | 39 | 0 | 0 |
| 2 | 500 | 0 | 0 |
| 3 | 500 | 0 | 0 |
| 4 | 500 | 7 | 0 |
| 5 | 500 | 33 | 20 |
| 6 | 500 | 72 | 68 |
| 7 | 500 | 132 | 103 |
| Total | 2500 | 244 | 191 |

Figure 4: Simulation results for a deconvolution network for a $K = 3$, rate 1/3 code. The network parameters were: $\lambda = 15$, $A = 6$, $B = 1$, $C = 0.45$, and $T = 0.025$.

For locally interconnected networks, the major concern is the flow of information through the network. In the simulations presented until now, the neurons in each stage are connected only to neurons in neighboring stages. A modified form of the network was also simulated in which the neurons in each stage are connected to the neurons in the four nearest neighboring stages. To implement this network, the subroutine to initialize the connection weights was modified to assign a non-zero value to $w_{i,j,i+2,k}$. This is straight-forward since, for a code with a constraint length of three, there is a single path connecting two nodes a distance two apart.

The results of this simulation are shown in the column labeled "four-nearest" in figure 4. It is easy to see that the network with the extra connections performs better

than the previous network. Most of the errors made by the nearest neighbor network occur for inputs in which the received subsequences $r_i$ and $r_{i+1}$ or $r_{i+2}$ contain a total of four or more errors. It appears that the network with the additional connections is, in effect, able to communicate around subsequences containing errors that block communications for the two-nearest neighbor network.

## 7 Summary and Conclusions

We have presented a locally interconnected network which minimizes a function that is analogous to the log likelihood function near the global minimum. The results of simulations demonstrate that the network can successfully decode input sequences containing no noise at least as well as the globally connected Hopfield-Tank [6] decomposition network. Simulations also strongly support the conjecture that in the noiseless case, the network can be guaranteed to converge to the global minimum. In addition, for low error rates, the network can also decode noisy received sequences.

We have been able to apply the Cohen-Grossberg proof of the stability of "on-center off-surround" networks to show that each stage will maximize the desired local "likelihood" for noisy received sequences. We have also shown that, in the large gain limit, the network as a whole is stable and that the equilibrium points correspond to the MLSE decoder output. Simulations have verified this proof of stability even for relatively small gains. Unfortunately, a proof of strict Lyapunov stability is very difficult, and may not be possible, because of the cooperative connections in the network.

This network demonstrates that it is possible to perform interesting functions even if only localized connections are allowed, although there may be some loss of performance. If we view the network as an associative memory, a trellis structured network that contains $NS$ neurons can correctly recall $2^S$ memories. Simulations of trellis networks strongly suggest that it is possible to guarantee a non-zero minimum radius of attraction for all memories. We are currently investigating the use of trellis structured layers in multilayer networks to explicitly provide the networks with the ability to tolerate errors and replace faulty neurons.

## References

[1] M. Cohen and S. Grossberg, "Absolute stability of global pattern formation and parallel memory storage by competitive neural networks," *IEEE Trans. Sys., Man, and Cyber.*, vol. 13, pp. 815–826, Sep.–Oct. 1983.

[2] S. Grossberg, "How does a brain build a cognitive code," in *Studies of Mind and Brain*, pp. 1–52, D. Reidel Pub. Co., 1982.

[3] J. Hopfield, "Neural networks and physical systems with emergent collective computational abilities," *Proceedings of the National Academy of Sciences USA*, vol. 79, pp. 2554–2558, 1982.

[4] J. Hopfield, "Neurons with graded response have collective computational properties like those of two-state neurons," *Proceedings of the National Academy of Science, USA*, vol. 81, pp. 3088–3092, May 1984.

[5] J. McClelland and D. Rumelhart, *Parallel Distributed Processing, Vol. 1*. The MIT Press, 1986.

[6] D. Tank and J. Hopfield, "Simple 'neural' optimization networks: an A/D converter, signal decision circuit and a linear programming circuit," *IEEE Trans. on Circuits and Systems*, vol. 33, pp. 533–541, May 1986.

[7] A. Viterbi and J. Omura, *Principles of Digital Communications and Coding*. McGraw-Hill, 1979.